# Prediction on Spike Data
# Using Kernel Algorithms

**Jan Eichhorn, Andreas Tolias, Alexander Zien, Malte Kuss,**
**Carl Edward Rasmussen, Jason Weston, Nikos Logothetis** and **Bernhard Schölkopf**
Max Planck Institute for Biological Cybernetics
72076 Tübingen, Germany
`first.last@tuebingen.mpg.de`

## Abstract

We report and compare the performance of different learning algorithms
based on data from cortical recordings. The task is to predict the orienta-
tion of visual stimuli from the activity of a population of simultaneously
recorded neurons. We compare several ways of improving the coding of
the input (i.e., the spike data) as well as of the output (i.e., the orienta-
tion), and report the results obtained using different kernel algorithms.

## 1   Introduction

Recently, there has been a great deal of interest in using the activity from a population
of neurons to predict or reconstruct the sensory input [1, 2], motor output [3, 4] or the
trajectory of movement of an animal in space [5]. This analysis is of importance since it
may lead to a better understanding of the coding schemes utilised by networks of neurons
in the brain. In addition, efficient algorithms to interpret the activity of brain circuits in
real time are essential for the development of successful brain computer interfaces such as
motor prosthetic devices.

The goal of reconstruction is to predict variables which can be of rather different nature
and are determined by the specific experimental setup in which the data is collected. They
might be for example arm movement trajectories or variables representing sensory stimuli,
such as orientation, contrast or direction of motion. From a data analysis perspective, these
problems are challenging for a number of reasons, to be discussed in the remainder of this
article.

We will exemplify our reasoning using data from an experiment described in Sect. 3. The
task is to reconstruct the angle of a visual stimulus, which can take eight discrete values,
from the activity of simultaneously recorded neurons.

**Input coding.** In order to effectively apply machine learning algorithms, it is essential to
adequately encode prior knowledge about the problem. A clever encoding of the input data
might reflect, for example, known invariances of the problem, or assumptions about the
similarity structure of the data motivated by scientific insights. An algorithmic approach
which currently enjoys great popularity in the machine learning community, called *kernel
machines*, makes these assumptions explicit by the choice of a kernel function. The ker-
nel can be thought of as a mathematical formalisation of a similarity measure that ideally

captures much of this prior knowledge about the data domain. Note that unlike many traditional machine learning methods, kernel machines can readily handle data that is not in the form of vectors of numbers, but also complex data types, such as strings, graphs, or spike trains. Recently, a kernel for spike trains was proposed whose design is based on a number of biologically motivated assumptions about the structure of spike data [6].

**Output coding.** Just like the inputs, also the stimuli perceived or the actions carried out by an animal are in general not given to us in vectorial form. Moreover, biologically meaningful similarity measures and *loss functions* may be very different from those used traditionally in pattern recognition. Hence, once again, there is a need for methods that are sufficiently general such that they can cope with these issues. In the problem at hand, the outputs are orientations of a stimulus and thus it would be desirable to use a method which takes their circular structure into account. In this paper, we will utilise the recently proposed kernel dependency estimation technique [7] that can cope with general sets of outputs and and a large class of loss functions in a principled manner. Besides, we also apply Gaussian process regression to the given task.

**Inference and generalisation.** The dimensionality of the spike data can be very high, in particular if the data stem from multicellular recording and if the temporal resolution is high. In addition, the problems are not necessarily stationary, the distributions can change over time, and depend heavily on the individual animal. These aspects make it hard for a learning machine to generalise from the training data to previously unseen *test* data. It is thus important to use methods which are state of the art and assay them using carefully designed numerical experiments. In our work, we have attempted to evaluate several such methods, including certain developments for the present task that shall be described below.

## 2   Learning algorithms, kernels and output coding

In supervised machine learning, we basically attempt to discover dependencies between variables based on a finite set of observations (called the training set) $\{(x_i, y_i)|i = 1, \ldots, n\}$. The $x_i \in X$ are referred to as inputs and are taken from a domain $X$; likewise, the $y \in Y$ are called outputs and the objective is to approximate the mapping $X \to Y$ between the domains from the samples. If $Y$ is a discrete set of class labels, e.g. $\{-1, 1\}$, the problem is referred to as classification; if $Y = \mathbb{R}^N$, it is called regression.

Kernel machines, a term which refers to a group of learning algorithms, are based on the notion of a feature space mapping $\Phi$. The input points get mapped to a possibly high-dimensional dot product space (called the feature space) using $\Phi$, and in that space the learning problem is tackled using simple linear geometric methods (see [8] for details). All geometric methods that are based on distances and angles can be performed in terms of the dot product. The "kernel trick" is to calculate the inner product of feature space mapped points using a kernel function

$$k(x_i, x_j) = \langle \Phi(x_i), \Phi(x_j) \rangle. \tag{1}$$

while avoiding explicit mappings $\Phi$. In order for $k$ to be interpretable as a dot product in some feature space it has to be a *positive definite* function.

### 2.1   Support Vector Classification and Gaussian Process Regression

A simple geometric classification method which is based on dot products and which is the basis of support vector machines is linear classification via separating hyperplanes. One can show that the so-called optimal separating hyperplane (the one that leads to the largest margin of separation between the classes) can be written in feature space as $\langle w, \Phi(x) \rangle + b = 0$, where the hyperplane normal vector can be expanded in terms of the training points as

$w = \sum_{i=1}^{m} \lambda_i \Phi(x_i)$. The points for which $\lambda_i \neq 0$ are called support vectors. Taken together, this leads to the decision function

$$f(x) = \text{sign}\left( \sum_{i=1}^{m} \lambda_i \langle \Phi(x), \Phi(x_i) \rangle + b \right) = \text{sign}\left( \sum_{i=1}^{m} \lambda_i k(x, x_i) + b \right). \qquad (2)$$

The coefficients $\lambda_i, b \in \mathbb{R}$ are found by solving a quadratic optimisation problem, for which standard methods exist. The central idea of support vector machines is thus that we can perform linear classification in a high-dimensional feature space using a kernel which can be seen as a (nonlinear) similarity measure for the input data. A popular nonlinear kernel function is the Gaussian kernel $k(x_i, x_j) = \exp(-\|x_i - x_j\|^2 / 2\sigma^2)$. This kernel has been successfully used to predict stimulus parameters using spikes from simultaneously recorded data [2].

In Gaussian process regression [9], the model specifies a random *distribution over functions*. This distribution is conditioned on the observations (the training set) and predictions may be obtained in closed form as Gaussian distributions for any desired test inputs. The characteristics (such as smoothness, amplitude, etc.) of the functions are given by the *covariance function* or *covariance kernel*; it controls how the outputs covary as a function of the inputs. In the experiments below (assuming $x \in \mathbb{R}^D$) we use a Gaussian kernel of the form

$$\text{Cov}(y_i, y_j) = k(x_i, x_j) = v^2 \exp\left( -\frac{1}{2} \sum_{d=1}^{D} \|x_i^d - x_j^d\|^2 / w_d^2 \right) \qquad (3)$$

with parameters $v$ and $\mathbf{w} = (w_1, \ldots, w_D)$. This covariance function expresses that outputs whose inputs are nearby have large covariance, and outputs that belong to inputs far apart have smaller covariance. In fact, it is possible to show that the distribution of functions generated by this covariance function are all smooth. The $\mathbf{w}$ parameters determine exactly how important different input coordinates are (and can be seen as a generalisation of the above kernel). The parameters are fit by optimising the likelihood.

## 2.2 Similarity measures for spike data

To take advantage of the strength of kernel machines in the analysis of cortical recordings we will explore the usefulness of different kernel functions. We describe the *spikernel* introduced in [6] and present a novel use of alignment-type scores typically used in bioinformatics.

Although we are far from understanding the neuronal code, there exist some reasonable assumptions about the structure of spike data one has to take into account when comparing spike patterns and designing kernels.

- Most fundamental is the assumption that frequency and temporal coding play central roles. Information related to a certain variable of the stimulus may be coded in highly specific temporal patterns contained in the spike trains of a cortical population.

- These firing patterns may be misaligned in time. To compare spike trains it might be necessary to realign them by introducing a certain time shift. We want the similarity score to be the higher the smaller this time shift is.

**Spikernel.** In [6] Shpigelman et al. proposed a kernel for spike trains that was designed with respect to the assumptions above and some extra assumptions related to the special task to be solved. To understand their ideas it is most instructive to have a look at the feature map $\Phi$ rather than at the kernel itself.

Let $\mathbf{s}$ be a sequence of firing rates of length $|\mathbf{s}|$. The feature map maps this sequence into a high dimensional space where the coordinates $\mathbf{u}$ represent a possible spike train prototype of fixed length $n \leq |\mathbf{s}|$. The value of the feature map of $\mathbf{s}$, $\Phi_{\mathbf{u}}(\mathbf{s})$, represents the similarity of $\mathbf{s}$ to the prototype $\mathbf{u}$. The $\mathbf{u}$ component of the feature vector $\Phi(\mathbf{s})$ is defined as:

$$\Phi_{\mathbf{u}}(\mathbf{s}) = C^{\frac{n}{2}} \sum_{\mathbf{i} \in \mathbf{I}_{n,|\mathbf{s}|}} \mu^{d(\mathbf{s_i}, \mathbf{u})} \lambda^{|\mathbf{s}| - \mathbf{i_1}} \tag{4}$$

Here $\mathbf{i}$ is an index vector that indexes a length $n$ ordered subsequence of $\mathbf{s}$ and the sum runs over all possible subsequences. $\lambda, \mu \in [0,1]$ are parameters of the kernel. The $\mu$-part of the sum reflects the weighting according to the similarity of $\mathbf{s}$ to the coordinate $\mathbf{u}$ (expressed in the distance measure $d(\mathbf{s_i}, \mathbf{u}) = \sum_{k=1}^{n} d(\mathbf{s_{i,k}}, \mathbf{u}_k)$), whereas the $\lambda$-part emphasises the concentration towards a "time of interest" at the end of the sequence $\mathbf{s}$ ($\mathbf{i}_1$ is the first index of the subsequence). Following the authors we chose the distance measure $d(\mathbf{s}_{i,k}, \mathbf{u}_k)$, determining how two firing rate vectors are compared, to be the squared $l_2$-norm: $d(\mathbf{s}_{i,k}, \mathbf{u}_k) = \|\mathbf{s}_{i,k} - \mathbf{u}_k\|_2^2$. Note, that each entry $\mathbf{s}_k$ of the sequence (-matrix) $\mathbf{s}$ is meant to be a vector containing the firing rates of all simultaneously recorded neurons in the same time interval (bin).

The kernel $k_n(\mathbf{s}, \mathbf{t})$ induced by this feature map can be computed in time $\mathcal{O}(|\mathbf{s}||\mathbf{t}|n)$ using dynamic programming. The kernel used in our experiments is a sum of kernels for different pattern lengths $n$ weighted with another parameter $p$, i.e.,
$k(\mathbf{s}, \mathbf{t}) = \sum_{i=1}^{N} p^i k_i(\mathbf{s}, \mathbf{t})$.

**Alignment score.** In addition to methods developed specifically for neural spike train data, we also train on pairwise similarities derived from global alignments. Aligning sequences is a standard method in bioinformatics; there, the sequences usually describe DNA, RNA or protein molecules. Here, the sequences are time-binned representations of the spike trains, as described above.

In a global alignment of two sequences $\mathbf{s} = s_1 \ldots s_{|s|}$ and $\mathbf{t} = t_1 \ldots t_{|t|}$, each sequence may be elongated by inserting copies of a special symbol (the dash, "_") at any position, yielding two stuffed sequences $\mathbf{s}'$ and $\mathbf{t}'$. The first requirement is that the stuffed sequences must have the same length. This allows to write them on top of each other, so that each symbol of $\mathbf{s}$ is either mapped to a symbol of $\mathbf{t}$ (match/mismatch), or mapped to a dash (gap), and vice versa. The second requirement for a valid alignment is that no dash is mapped to a dash, which restricts the length of any alignment to a maximum of $|\mathbf{s}| + |\mathbf{t}|$.

Once costs are assigned to the matches and gaps, the cost of an alignment is defined as the sum of costs in the alignment. The distance of $\mathbf{s}$ and $\mathbf{t}$ can now be defined as the cost of an optimal global alignment of $\mathbf{s}$ and $\mathbf{t}$, where optimal means minimising the cost. Although there are exponentially many possible global alignments, the optimal cost (and an optimal alignment) can be computed in time $\mathcal{O}(|\mathbf{s}||\mathbf{t}|)$ using dynamic programming [10].

Let $c(a, b)$ denote the cost of a match/mismatch ($a = s_i, b = t_j$) or of a gap (either $a =$ "_" or $b =$ "_"). We parameterise the costs with $\gamma$ and $\mu$ as follows:

$$\begin{aligned} c(a,b) = c(b,a) &:= |a - b| \\ c(a,\_) = c(\_,a) &:= \gamma|a - \mu| \end{aligned}$$

The matrix of pairwise distances as defined above will, in general, not be a proper kernel (i.e., it will not be positive definite). Therefore, we use it to build a new representation of the data (see below). A related but different distance measure has previously been proposed by Victor and Purpura [11].

We use the alignment score to compute explicit feature vectors of the data points via an

empirical kernel map [8, p. 42]. Consider as prototypes the *overall* data set[1] $\{\mathbf{x}_i\}_{i=1,...,m}$ of $m$ trials $\mathbf{x}_i = [\mathbf{n}_{1,i} \ \mathbf{n}_{2,i} \ ... \ \mathbf{n}_{20,i}]$ as defined in Sect. 3. Since our alignment score $k_{align}(\mathbf{n}, \mathbf{n}')$ applies to single spike trains only[2], we compute the empirical kernel map for each neuron separately and then concatenate these vectors. Hence, the feature map is defined as:

$$\Phi_{\mathbf{x}_1,...,\mathbf{x}_m}(\mathbf{x}') = \Phi_{\mathbf{x}_1,...,\mathbf{x}_m}([\mathbf{n}'_1 \ \mathbf{n}'_2 \ ... \ \mathbf{n}'_{20}])$$
$$= [\{k_{align}(\mathbf{n}_{1,i=1..m}, \mathbf{n}'_1)\} \ \{k_{align}(\mathbf{n}_{2,i=1..m}, \mathbf{n}'_2)\} ... \{k_{align}(\mathbf{n}_{20,i=1..m}, \mathbf{n}'_{20})\}]$$

Thus, each trial is represented by a vector of its alignment score with respect to all other trials where alignments are computed separately for all 20 neurons.

We can now train kernel machines using any standard kernel on top of this representation, but we already achieve very good performance using the simple linear kernel (see results section). Although we give results obtained with this technique of constructing a feature map only for the alignment score, it can be easily applied with the spikernel and other kernels.

### 2.3 Coding structure in output space

Our objective is to use various machine learning algorithms to predict the orientation of a stimulus used in the experiment described below. Since we use discrete orientations we can model this as a multi-class classification problem or transform it into a regression task.

**Combining Support Vector Machines.** Above, we explained how to do binary classification using SVMs by estimating a normal vector $w$ and offset $b$ of a hyperplane $\langle w, \Phi(x) \rangle + b = 0$ in the feature space. A given point $x$ will then be assigned to class 1 if $\langle w, \Phi(x) \rangle + b > 0$ (and to class -1 otherwise). If we have $M > 2$ classes, we can train $M$ classifiers, each one separating one specific class from the union of all other ones (hence the name "one-versus-rest"). When classifying a new point $x$, we simply assign it to the class whose classifier leads to the largest value of $\langle w, \Phi(x) \rangle + b$.
A more sophisticated and more expensive method is to train one classifier for each possible combination of two classes and then use a voting scheme to classify a point. It is referred to as "one-versus-one".

**Kernel Dependency Estimation.** Note that the above approach treats all classes the same. In our situation, however, certain classes are "closer" to each other since the corresponding stimulus angles are closer than others. To take this into account, we use the kernel dependency estimation (KDE) algorithm [7] with an output similarity measure corresponding to a loss function of the angles taking the form $L(\alpha, \beta) = \cos(2\alpha - 2\beta)$.[3] The modification respects the symmetry that $0°$ and $180°$, say, are equivalent.

Lack of space does not permit us to explain the KDE algorithm in detail. In a nutshell, it estimates a linear mapping between two feature spaces. One feature space corresponds to the kernel used on the inputs (in our case, the spike trains), and the other one to a second kernel which encodes the similarity measure to be used on the outputs (the orientation of the lines).

**Gaussian Process Regression.** When we use Gaussian processes to predict the stimulus angle $\alpha$ we consider the task as a regression problem on $\sin 2\alpha$ and $\cos 2\alpha$ separately. To

do prediction we take the means of the predicted distributions of $\sin 2\alpha$ and $\cos 2\alpha$ as point estimates respectively, which are then projected onto the unit circle. Finally we assign the averaged predicted angle to the nearest orientation which could have been shown.

## 3 Experiments

We will now apply the ideas from the reasoning above and see how well these different concepts perform in practice on a dataset of cortical recordings.

**Data collection.** The dataset we used was collected in an experiment performed in our neurophysiology department. All experiments were conducted in full compliance with the guidelines of the European Community (EUVD/86/609/EEC) for the care and use of laboratory animals and were approved by the local authorities (Regierungspräsidium). The spike data were recorded using tetrodes inserted in area V1 of a behaving macaque (Macaca Mulatta). The spike waveforms were sampled at 32KHz. The animal's task was to fixate a small square spot on the monitor while gratings of eight different orientations ($0^o$, $22^o$, $45^o$, $67^o$, $90^o$, $112^o$, $135^o$, $158^o$) and two contrasts (2% and 30%) were presented on a monitor. The stimuli were positioned on the monitor so as to cover the classical receptive fields of the neurons. A single stimulus of fixed orientation and contrast was presented for a period of 500 ms, i.e., during the epoch of a single behavioural trial. All 8 stimuli appeared 30 times each and in random order, resulting in 240 observed trials.

Spiking activity from neural recordings usually come as a time series of action potentials from one or more neurons recorded from the brain. It is commonly believed that in most circumstances most of the information in the spiking activity is mainly present in the times of occurrence of spikes and not in the exact shape of the individual spikes. Therefore we can abstract the spike series as a series of zeros and ones.

From a single trial we have recordings of $500ms$ from 20 neurons. We compute the firing rates from the high resolution data for each neuron in 1, 5 or 10 bins of length 500, 100 or $50ms$ respectively, resulting in three different data representations for different temporal resolutions. By concatenation of the vectors $\mathbf{n}_r$ ($r = 1, \ldots, 20$) containing the bins of each neuron we obtain one data point $\mathbf{x} = [\mathbf{n}_1 \ \mathbf{n}_2 \ ... \ \mathbf{n}_{20}]$ per trial.

**Comparing the algorithms.** Below we validate our reasoning on input and output coding with several experiments. We will compare the kernel algorithms KDE, SVM and Gaussian Processes (GP) and a simple k-nearest neighbour approach (k-NN) that we applied with different kernels and different data representations. As reference values, we give the performance of a standard Bayesian reconstruction method (assuming independent neurons with Poisson characteristics), a Template Matching method and the standard Population Vector method as they are described e.g. in [5] and [3].

In all our experiments we compute the test error over a five fold cross-validation using always the same data split, balanced with respect to the classes.[4] We use four out of the five folds of the data to choose the parameters of the kernel and the method. This choice itself is done via another level of five fold cross-validation (this time unbalanced). Finally we train the best model on these four folds and compute an independent test error on the remaining fold.

Since simple zero-one-loss is not very informative about the error in multi-class problems, we report the linear loss of the predicted angles, while taking into account the circular structure of the problem. Hence the loss function takes the form

$$L(\alpha, \beta) = \min\{|\alpha - \beta|, -|\alpha - \beta| + 180^o\}. \tag{5}$$

The parameters of the KDE algorithm (ridge parameter) and the SVM ($C$) are taken from a logarithmic grid ($ridge = 10^{-5}, 10^{-4}, ..., 10^{1}$; $C = 10^{-1}, 1, ..., 10^{5}$). After we knew its order of magnitude, we chose the $\sigma$-parameter of the Gaussian kernel from a linear grid ($\sigma = 1, 2, ..., 10$). The spikernel has four parameters: $\lambda$, $\mu$, $N$ and $p$. The stimulus in our experiment was perceived over the whole period of recording. Therefore we do not want any increasing weight of the similarity score towards the beginning or the end of the spike-sequence and we fix $\lambda = 1$. Further we chose $N = 10$ to be the length of our sequence, and thereby consider patterns of all possible lengths. The parameters $\mu$ and $p$ are chosen from the following (partly linear) grids: $\mu = 0.01, 0.05, 0.1, 0.2, 0.3, 0.4, ..., 0.8, 0.9, 0.99$ and $p = 0.05, 0.1, 0.3, 0.5, ..., 2.5, 2.7$

**Table 1** Mean test error and standard error on the low contrast dataset

|  |  | Gaussian Kernel | Spikernel | Alignment score |
|---|---|---|---|---|
| KDE | 10 bins | $16.8° \pm 1.6°$ | $\mathbf{11.5° \pm 1.3°}$ | $13.8° \pm 1.3°$ |
|  | 1 bin | $12.8° \pm 1.7°$ | $(13.6° \pm 1.8°)^{\dagger}$ |  |
| SVM (1-vs-rest) | 10 bins | $16.8° \pm 2.0°$ | $13.1° \pm 1.4°$ | $12.8° \pm 0.9°$ |
|  | 1 bin | $13.3° \pm 1.6°$ |  |  |
| SVM (1-vs-1) | 10 bins | $16.4° \pm 1.6°$ | $\mathbf{11.2° \pm 1.3°}$ | $\mathbf{12.3° \pm 1.5°}$ |
|  | 1 bin | $\mathbf{12.2° \pm 1.7°}$ |  |  |
| k-NN | 10 bins | $18.7° \pm 1.5°$ | $\mathbf{12.1° \pm 1.4°}$ | $13.0° \pm 2.0°$ |
|  | 1 bin | $14.0° \pm 1.7°$ |  |  |
| GP | 2 bins $^{\ddagger}$ | $16.2° \pm 1.1°$ | n/a $^{*}$ | n/a $^{*}$ |
|  | 1 bin | $15.6° \pm 1.7°$ |  |  |

Bayesian rec.: $14.4° \pm 2.1°$, Template Matching: $17.7° \pm 0.6°$, Pop. Vect.: $28.8° \pm 1.0°$

**Table 2** Mean test error and standard error on the high contrast dataset

|  |  | Gaussian Kernel | Spikernel | Alignment score |
|---|---|---|---|---|
| KDE | 10 bins | $1.9° \pm 0.5°$ | $1.7° \pm 0.4°$ | $2.1° \pm 0.4°$ |
|  | 1 bin | $1.4° \pm 0.5°$ | $(1.6° \pm 0.4°)^{\dagger}$ |  |
| SVM (1-vs-rest) | 10 bins | $1.5° \pm 0.5°$ | $1.4° \pm 0.6°$ | $\mathbf{1.0° \pm 0.5°}$ |
|  | 1 bin | $1.4° \pm 0.4°$ |  |  |
| SVM (1-vs-1) | 10 bins | $1.2° \pm 0.4°$ | $1.4° \pm 0.5°$ | $\mathbf{0.8° \pm 0.3°}$ |
|  | 1 bin | $\mathbf{1.1° \pm 0.4°}$ |  |  |
| k-NN | 10 bins | $4.7° \pm 1.2°$ | $\mathbf{1.0° \pm 0.4°}$ | $\mathbf{1.0° \pm 0.3°}$ |
|  | 1 bin | $1.7° \pm 0.6°$ |  |  |
| GP | 2 bins $^{\ddagger}$ | $1.4° \pm 0.4°$ | n/a $^{*}$ | n/a $^{*}$ |
|  | 1 bin | $2.0° \pm 0.5°$ |  |  |

Bayesian rec.: $3.8° \pm 0.6°$, Template Matching: $7.2° \pm 1.0°$, Pop. Vect.: $11.6° \pm 0.7°$

$^{\dagger}$ We report this number only for comparison, since the spikernel relies on temporal patterns and it makes no sense to use only one bin.
$^{\ddagger}$ A 10 bin resolution would require to determine 200 parameters $w_d$ of the covariance function (3) from only 192 samples.
$^{*}$ We did not compute these results. Both kernels are not analytical functions of their parameters and we would loose much of the convenience of Gaussian Processes. Using crossvalidation instead resembles very much Kernel Ridge Regression on $\sin 2\alpha$ and $\cos 2\alpha$ which is almost exactly what KDE is doing when applied with the loss function (5).

The results for the low contrast datasets is given in Table 1, and Table 2 presents results for high contrast (five best results in boldface). The relatively large standard error ($\pm \frac{\sigma}{\sqrt{n}}$) is due to the fact that we used only five folds to compute the test error.

## 4  Discussion

In our experiments, we have shown that using modern machine learning techniques, it is possible to use tetrode recordings in area V1 to reconstruct the orientation of a stimulus presented to a macaque monkey rather accurately: depending on the contrast of the stimulus, we obtained error rates in the range of $1° - 20°$. We can observe that standard techniques for decoding, namely Population vector, Template Matching and a particular Bayesian reconstruction method, can be outperformed by state-of-the-art kernel methods when applied with an appropriate kernel and suitable data representation. We found that the accuracy of kernel methods can in most cases be improved by utilising task specific similarity measures for spike trains, such as the spikernel or the introduced alignment distances from bioinformatics. Due to the (by machine learning standards) relatively small size of the analysed datasets, it is hard to draw conclusions regarding which of the applied kernel methods performs best.

Rather than focusing too much on the differences in performance, we want to emphasise the capability of kernel machines to assay different decoding hypotheses by choosing appropriate kernel functions. Analysing their respective performance may provide insight about how spike trains carry information and thus about the nature of neural coding.

**Acknowledgements.**  For useful help, we thank Goekhan Bakır, Olivier Bousquet and Gunnar Rätsch. J.E. was supported by a grant from the Studienstiftung des deutschen Volkes.

## Footnotes

[1]Note that this means that we are considering a transductive setting [12], where we have access to all input data (but not the test outputs) during training.

[2]It is straightforward to extend this idea to synchronous alignments of the whole population vector, but we achieved worse results.

[3]Note that $L(\alpha, \beta)$ needs to be an admissible kernel, i.e. positive definite, and therefore we cannot use the linear loss function (5).

[4] I.e., in every fold we have the same number of points per class.

## References

[1]  P. Földiák. The "ideal humunculus": statistical inference from neural population responses. In F. Eeckman and J. Bower, editors, *Computation and Neural Systems 1992*, Norwell, MA, 1993. Kluwer.

[2]  A. S. Tolias, A. G. Siapas, S. M. Smirnakis and N. K. Logothetis. Coding visual information at the level of populations of neurons. *Soc. Neurosci. Abst. 28*, 2002.

[3]  A. P. Georgopoulos, A. B. Schwartz and R. E. Kettner. Neuronal population coding of movement direction. *Science*, 233(4771):1416–1419, 1986.

[4]  T. D. Sanger. Probability density estimation for the interpretation of neural population codes. *J Neurophysiol.*, 76(4):2790–2793, 1996.

[5]  K. Zhang, I. Ginzburg, B. L. McNaughton and T. J. Sejnowski. Interpreting neuronal population activity by reconstruction: unified framework with application to hippocampal place cells. *J Neurophysiol.*, 79(2):1017–1044, 1998.

[6]  L. Shpigelman, Y. Singer, R. Paz and E. Vaadia. Spikernels: embedding spike neurons in inner-product spaces. In S. Becker, S. Thrun and K. Obermayer, editors, *Advances in Neural Information Processing Systems 15*, 2003.

[7]  J. Weston, O. Chapelle, A. Elisseeff, B. Schölkopf and V. Vapnik. Kernel dependency estimation. In S. Becker, S. Thrun and K. Obermayer, editors, *Advances in Neural Information Processing Systems 15*, 2003.

[8]  B. Schölkopf and A. J. Smola. *Learning with Kernels*. The MIT Press, Cambridge, Massachusetts, 2002.

[9]  C. K. I. Williams and C. E. Rasmussen. Gaussian processes for regression. In D. S. Touretzky, M. C. Mozer and M. E. Hasselmo, editors, *Advances in Neural Information Processing Systems 8*, 1996.

[10]  S. B. Needleman and C. D. Wunsch. A General Method Applicable to the Search for Similarities in the Amino Acid Sequence of Two Proteins. *Journal of Molecular Biology*, 48:443–453, 1970.

[11]  J. D. Victor and K. P. Purpura. Nature and precision of temporal coding in visual cortex: a metric-space analysis. *J Neurophysiol*, 76(2):1310–1326, 1996.

[12]  V. N. Vapnik. *Statistical Learning Theory*. John Wiley & Sons, New York, 1998.